# Mind the Duality Gap:
# Logarithmic regret algorithms for online optimization

**Sham M. Kakade**
Toyota Technological Institute at Chicago
sham@tti-c.org

**Shai Shalev-Shwartz**
Toyota Technological Institute at Chicago
shai@tti-c.org

## Abstract

We describe a primal-dual framework for the design and analysis of online strongly convex optimization algorithms. Our framework yields the tightest known logarithmic regret bounds for Follow-The-Leader and for the gradient descent algorithm proposed in Hazan et al. [2006]. We then show that one can interpolate between these two extreme cases. In particular, we derive a new algorithm that shares the computational simplicity of gradient descent but achieves lower regret in many practical situations. Finally, we further extend our framework for generalized strongly convex functions.

## 1 Introduction

In recent years, online regret minimizing algorithms have become widely used and empirically successful algorithms for many machine learning problems. Notable examples include efficient learning algorithms for structured prediction and ranking problems [Collins, 2002, Crammer et al., 2006]. Most of these empirically successful algorithms are based on algorithms which are tailored to general convex functions, whose regret is $O(\sqrt{T})$. Rather recently, there is a growing body of work providing online algorithms for *strongly* convex loss functions, with regret guarantees that are only $O(\log T)$. These algorithms have potential to be highly applicable since many machine learning optimization problems are in fact strongly convex — either with strongly convex loss functions (e.g. log loss, square loss) or, indirectly, via strongly convex regularizers (e.g. $L_2$ or $KL$ based regularization). Note that in this later case, the loss function itself may only be just convex but a strongly convex regularizer effectively makes this a strongly convex optimization problem (e.g. the SVM optimization problem uses the hinge loss with $L_2$ regularization). The aim of this paper is to provide a template for deriving a wider class of regret-minimizing algorithms for online strongly convex programming.

Online convex optimization takes place in a sequence of consecutive rounds. At each round, the learner predicts a vector $\mathbf{w}_t \in S \subset \mathbb{R}^n$, and the environment responds with a convex loss function, $\ell_t : S \to \mathbb{R}$. The goal of the learner is to minimize the difference between his cumulative loss and the cumulative loss of the optimal fixed vector, $\sum_{t=1}^{T} \ell_t(\mathbf{w}_t) - \min_{\mathbf{w} \in S} \sum_{t=1}^{T} \ell_t(\mathbf{w})$. This is termed 'regret' since it measures how 'sorry' the learner is, in retrospect, not to have predicted the optimal vector.

Roughly speaking, the family of regret minimizing algorithms (for general convex functions) can be seen as varying on two axes, the 'style' and the 'aggressiveness' of the update. In addition to online algorithms' relative simplicity, the empirical successes are also due to having these two knobs to tune for the problem at hand (which determine the nature of the regret bound). By style, we mean updates which favor either rotational invariance (such as gradient descent like update rules) or sparsity (like the multiplicative updates). Of course there is a much richer family here, including the $L_p$ updates. By the aggressiveness of the update, we mean how much the algorithm moves its decision to be consistent with most recent loss functions. For example, the preceptron algorithm makes no update

when there is no error. In contrast, there is a family of algorithms which more aggressively update the loss when there is a margin mistake. These algorithms are shown to have improved performance (see for example the experimental study in Shalev-Shwartz and Singer [2007b]).

While historically much of the analysis of these algorithms have been done on a case by case basis, in retrospect, the proof techniques have become somewhat boilerplate, which has lead to growing body of work to unify these analyses (see Cesa-Bianchi and Lugosi [2006] for review). Perhaps the most unified view of these algorithms is the 'primal-dual' framework of Shalev-Shwartz and Singer [2006], Shalev-Shwartz [2007], for which the gamut of these algorithms can be largely viewed as special cases. Two aspects are central in providing this unification. First, the framework works with a complexity function, which determines the style of algorithm and the nature of the regret guarantee (If this function is the $L_2$ norm, then one obtains gradient like updates, and if this function is the $KL$-distance, then one obtains multiplicative updates). Second, the algorithm maintains both "primal" and "dual" variables. Here, the the primal objective function is $\sum_{t=1}^{T} \ell_t(w)$ (where $\ell_t$ is the loss function provided at round $t$), and one can construct a dual objective function $\mathcal{D}_t(\cdot)$, which only depends on the loss functions $\ell_1, \ell_2, \ldots \ell_{t-1}$. The algorithm works by incrementally increasing the dual objective value (in an online manner), which can be done since each $\mathcal{D}_t$ is only a function of the previous loss functions. By weak duality, this can be seen as decreasing the duality gap. The level of aggressiveness is seen to be how fast the algorithm is attempting to increase the dual objective value.

This paper focuses on extending the duality framework for online convex programming to the case of strongly convex functions. This analysis provides a more unified and intuitive view of the extant algorithms for online strongly convex programming. An important observation we make is that any $\sigma$-strongly convex loss function can be rewritten as $\ell_i(\mathbf{w}) = f(\mathbf{w}) + g_i(\mathbf{w})$, where $f$ is a fixed $\sigma$-strongly convex function (i.e. $f$ does not depend on $i$), and $g_i$ is a convex function. Therefore, after $t$ online rounds, the amount of intrinsic strong convexity we have in the primal objective $\sum_{i=1}^{t} \ell_t(\mathbf{w})$ is at least $\sigma t$. In particular, this explains the learning rate of $\frac{1}{\sigma t}$ proposed in the gradient descent algorithm of Hazan et al. [2006]. Indeed, we show that our framework includes the gradient descent algorithm of Hazan et al. [2006] as an important special case, in which the aggressiveness level is minimal. At the most aggressive end, our framework yields the Follow-The-Leader algorithm. Furthermore, the template algorithm serves as a vehicle for deriving new algorithms (which enjoy logarithmic regret guarantees).

The remainder of the paper is outlined as follows. We first provide background on convex duality. As a warmup, in Section 3, we present an intuitive primal-dual analysis of Follow-The-Leader (FTL), when $f$ is the Euclidean norm. This naturally leads to a more general primal-dual algorithm (for which FTL is a special case), which we present in Section 4. Next, we further generalize our algorithmic framework to include strongly convex complexity functions $f$ with respect to arbitrary norms $\|\cdot\|$. We note that the introduction of a complexity function was already provided in Shalev-Shwartz and Singer [2007a], but the analysis is rather specialized and does not have a knob which can tune the aggressiveness of the algorithm. Finally, in Sec. 6 we conclude with a side-by-side comparison of our algorithmic framework for strongly convex functions and the framework for (non-strongly) convex functions given in Shalev-Shwartz [2007].

## 2 Mathematical Background

We denote scalars with lower case letters (e.g. $w$ and $\lambda$), and vectors with bold face letters (e.g. $\mathbf{w}$ and $\boldsymbol{\lambda}$). The inner product between vectors $\mathbf{x}$ and $\mathbf{w}$ is denoted by $\langle \mathbf{x}, \mathbf{w} \rangle$. To simplify our notation, given a sequence of vectors $\boldsymbol{\lambda}_1, \ldots, \boldsymbol{\lambda}_t$ or a sequence of scalars $\sigma_1, \ldots, \sigma_t$ we use the shorthand

$$\boldsymbol{\lambda}_{1:t} = \sum_{i=1}^{t} \boldsymbol{\lambda}_i \quad \text{and} \quad \sigma_{1:t} = \sum_{i=1}^{t} \sigma_i \ .$$

Sets are designated by upper case letters (e.g. $S$). The set of non-negative real numbers is denoted by $\mathbb{R}_+$. For any $k \geq 1$, the set of integers $\{1, \ldots, k\}$ is denoted by $[k]$. A norm of a vector $\mathbf{x}$ is denoted by $\|\mathbf{x}\|$. The dual norm is defined as $\|\boldsymbol{\lambda}\|_\star = \sup\{\langle \mathbf{x}, \boldsymbol{\lambda} \rangle : \|\mathbf{x}\| \leq 1\}$. For example, the Euclidean norm, $\|\mathbf{x}\|_2 = (\langle \mathbf{x}, \mathbf{x} \rangle)^{1/2}$ is dual to itself and the $L_1$ norm, $\|\mathbf{x}\|_1 = \sum_i |x_i|$, is dual to the $L_\infty$ norm, $\|\mathbf{x}\|_\infty = \max_i |x_i|$.

$$
\boxed{
\begin{aligned}
&\text{FOR } t = 1, 2, \ldots, T: \\
&\quad \text{Define } \mathbf{w}_t = -\tfrac{1}{\sigma_{1:(t-1)}} \boldsymbol{\lambda}^t_{1:(t-1)} \\
&\quad \text{Receive a function } \ell_t(\mathbf{w}) = \tfrac{\sigma_t}{2}\|\mathbf{w}\|^2 + g_t(\mathbf{w}) \text{ and suffer loss } \ell_t(\mathbf{w}_t) \\
&\quad \text{Update } \boldsymbol{\lambda}^{t+1}_1, \ldots, \boldsymbol{\lambda}^{t+1}_t \text{ s.t. the following holds} \\
&\qquad (\boldsymbol{\lambda}^{t+1}_1, \ldots, \boldsymbol{\lambda}^{t+1}_t) \in \operatorname*{argmax}_{\boldsymbol{\lambda}_1, \ldots, \boldsymbol{\lambda}_t} \mathcal{D}_{t+1}(\boldsymbol{\lambda}_1, \ldots, \boldsymbol{\lambda}_t)
\end{aligned}
}
$$

Figure 1: A primal-dual view of Follow-the-Leader. Here the algorithm's decision $\mathbf{w}_t$ is the best decision with respect to the previous losses. This presentation exposes the implicit role of the dual variables. Slightly abusing notation, $\boldsymbol{\lambda}_{1:0} = 0$, so that $\mathbf{w}_1 = 0$. See text.

We next recall a few definitions from convex analysis. A function $f$ is $\sigma$-*strongly* convex if

$$
f(\alpha\mathbf{u} + (1-\alpha)\mathbf{v}) \le \alpha f(\mathbf{u}) + (1-\alpha)f(\mathbf{v}) - \frac{\sigma}{2}\alpha(1-\alpha)\|\mathbf{u} - \mathbf{v}\|^2_2 .
$$

In Sec. 5 we generalize the above definition to arbitrary norms. If a function $f$ is $\sigma$-strongly convex then the function $g(\mathbf{w}) = f(\mathbf{w}) - \frac{\sigma}{2}\|\mathbf{w}\|^2$ is convex.

The Fenchel conjugate of a function $f : S \to \mathbb{R}$ is defined as

$$
f^\star(\boldsymbol{\theta}) = \sup_{\mathbf{w}\in S} \langle \mathbf{w}, \boldsymbol{\theta}\rangle - f(\mathbf{w}) .
$$

If $f$ is closed and convex, then the Fenchel conjugate of $f^\star$ is $f$ itself (a function is closed if for all $\alpha > 0$ the level set $\{\mathbf{w} : f(\mathbf{w}) \le \alpha\}$ is a closed set). It is straightforward to verify that the function $f(\mathbf{w})$ is conjugate to itself. The definition of $f^\star$ also implies that for $c > 0$ we have $(c\,f)^\star(\boldsymbol{\theta}) = c\,f^\star(\boldsymbol{\theta}/c)$.

A vector $\boldsymbol{\lambda}$ is a sub-gradient of a function $f$ at $\mathbf{w}$ if for all $\mathbf{w}' \in S$, we have that $f(\mathbf{w}') - f(\mathbf{w}) \ge \langle \mathbf{w}' - \mathbf{w}, \boldsymbol{\lambda}\rangle$. The differential set of $f$ at $\mathbf{w}$, denoted $\partial f(\mathbf{w})$, is the set of all sub-gradients of $f$ at $\mathbf{w}$. If $f$ is differentiable at $\mathbf{w}$, then $\partial f(\mathbf{w})$ consists of a single vector which amounts to the gradient of $f$ at $\mathbf{w}$ and is denoted by $\nabla f(\mathbf{w})$.

The Fenchel-Young inequality states that for any $\mathbf{w}$ and $\boldsymbol{\theta}$ we have that $f(\mathbf{w}) + f^\star(\boldsymbol{\theta}) \ge \langle \mathbf{w}, \boldsymbol{\theta}\rangle$. Sub-gradients play an important role in the definition of the Fenchel conjugate. In particular, the following lemma, whose proof can be found in Borwein and Lewis [2006], states that if $\boldsymbol{\lambda} \in \partial f(\mathbf{w})$ then the Fenchel-Young inequality holds with equality.

**Lemma 1** *Let $f$ be a closed and convex function and let $\partial f(\mathbf{w}')$ be its differential set at $\mathbf{w}'$. Then, for all $\boldsymbol{\lambda}' \in \partial f(\mathbf{w}')$, we have $f(\mathbf{w}') + f^\star(\boldsymbol{\lambda}') = \langle \boldsymbol{\lambda}', \mathbf{w}'\rangle$ .*

We make use of the following variant of Fenchel duality (see the appendix for more details):

$$
\max_{\boldsymbol{\lambda}_1, \ldots, \boldsymbol{\lambda}_T} -f^\star\!\left(-\sum_{t=1}^T \boldsymbol{\lambda}_t\right) - \sum_{t=1}^T g_t^\star(\boldsymbol{\lambda}_t) \le \min_{\mathbf{w}} f(\mathbf{w}) + \sum_{t=1}^T g_t(\mathbf{w}) . \tag{1}
$$

## 3 Warmup: A Primal-Dual View of Follow-The-Leader

In this section, we provide a dual analysis for the FTL algorithm. The dual view of FTL will help us to derive a family of logarithmic regret algorithms for online convex optimization with strongly convex functions.

Recall that FTL algorithm is defined as follows:

$$
\mathbf{w}_t = \operatorname*{argmin}_{\mathbf{w}} \sum_{i=1}^{t-1} \ell_i(\mathbf{w}) . \tag{2}
$$

For each $i \in [t-1]$ define $g_i(\mathbf{w}) = \ell_i(\mathbf{w}) - \frac{\sigma_i}{2}\|\mathbf{w}\|^2$, where $\sigma_i$ is the largest scalar such that $g_i$ is still a convex function. The assumption that $\ell_i$ is $\sigma$-strongly convex guarantees that $\sigma_i \ge \sigma$. We can

therefore rewrite the objective function on the right-hand side of Eq. (2) as

$$\mathcal{P}_t(\mathbf{w}) \;=\; \frac{\sigma_{1:(t-1)}}{2}\|\mathbf{w}\|^2 + \sum_{i=1}^{t-1} g_i(\mathbf{w}) \;\;, \tag{3}$$

(recall that $\sigma_{1:(t-1)} = \sum_{i=1}^{t-1}\sigma_i$). The Fenchel dual optimization problem (see Sec. 2) is to maximize the following dual objective function

$$\mathcal{D}_t(\boldsymbol{\lambda}_1,\ldots,\boldsymbol{\lambda}_{t-1}) \;=\; -\frac{1}{2\,\sigma_{1:(t-1)}}\|\boldsymbol{\lambda}_{1:(t-1)}\|^2 - \sum_{i=1}^{t-1} g_i^\star(\boldsymbol{\lambda}_i) \;\;. \tag{4}$$

Let $(\boldsymbol{\lambda}_1^t,\ldots,\boldsymbol{\lambda}_{t-1}^t)$ be the maximizer of $\mathcal{D}_t$. The relation between the optimal dual variables and the optimal primal vector is given by (see again Sec. 2)

$$\mathbf{w}_t \;=\; -\frac{1}{\sigma_{1:(t-1)}}\boldsymbol{\lambda}_{1:(t-1)}^t \;\;. \tag{5}$$

Throughout this section we assume that strong duality holds (i.e. Eq. (1) holds with equality). See the appendix for sufficient conditions. In particular, under this assumption, we have that the above setting for $\mathbf{w}_t$ is in fact a minimizer of the primal objective, since $(\boldsymbol{\lambda}_1^t,\ldots,\boldsymbol{\lambda}_{t-1}^t)$ maximizes the dual objective (see the appendix). The primal-dual view of Follow-the-Leader is presented in Figure 1.

Denote

$$\Delta_t \;=\; \mathcal{D}_{t+1}(\boldsymbol{\lambda}_1^{t+1},\ldots,\boldsymbol{\lambda}_t^{t+1}) - \mathcal{D}_t(\boldsymbol{\lambda}_1^t,\ldots,\boldsymbol{\lambda}_{t-1}^t) \;. \tag{6}$$

To analyze the FTL algorithm, we first note that (by strong duality)

$$\sum_{t=1}^{T}\Delta_t \;=\; \mathcal{D}_{T+1}(\boldsymbol{\lambda}_1^{T+1},\ldots,\boldsymbol{\lambda}_T^{T+1}) \;=\; \min_{\mathbf{w}}\mathcal{P}_{T+1}(\mathbf{w}) \;=\; \min_{\mathbf{w}}\sum_{t=1}^{T}\ell_t(\mathbf{w}) \;. \tag{7}$$

Second, the fact that $(\boldsymbol{\lambda}_1^{t+1},\ldots,\boldsymbol{\lambda}_t^{t+1})$ is the maximizer of $\mathcal{D}_{t+1}$ implies that for any $\boldsymbol{\lambda}$ we have

$$\Delta_t \;\geq\; \mathcal{D}_{t+1}(\boldsymbol{\lambda}_1^t,\ldots,\boldsymbol{\lambda}_{t-1}^t,\boldsymbol{\lambda}) - \mathcal{D}_t(\boldsymbol{\lambda}_1^t,\ldots,\boldsymbol{\lambda}_{t-1}^t) \;. \tag{8}$$

The following central lemma shows that there exists $\boldsymbol{\lambda}$ such that the right-hand side of the above is sufficiently large.

**Lemma 2** *Let $(\boldsymbol{\lambda}_1,\ldots,\boldsymbol{\lambda}_{t-1})$ be an arbitrary sequence of vectors. Denote $\mathbf{w} = -\frac{1}{\sigma_{1:(t-1)}}\boldsymbol{\lambda}_{1:(t-1)}$, let $\mathbf{v} \in \partial\ell_t(\mathbf{w})$, and let $\boldsymbol{\lambda} = \mathbf{v} - \sigma_t\mathbf{w}$. Then, $\boldsymbol{\lambda} \in \partial g_t(\mathbf{w})$ and*

$$\mathcal{D}_{t+1}(\boldsymbol{\lambda}_1,\ldots,\boldsymbol{\lambda}_{t-1},\boldsymbol{\lambda}) - \mathcal{D}_t(\boldsymbol{\lambda}_1,\ldots,\boldsymbol{\lambda}_{t-1}) \;=\; \ell_t(\mathbf{w}) - \frac{\|\mathbf{v}\|^2}{2\,\sigma_{1:t}} \;.$$

**Proof** We prove the lemma for the case $t > 1$. The case $t = 1$ can be proved similarly. Since $\ell_t(\mathbf{w}) = \frac{\sigma_t}{2}\|\mathbf{w}\|^2 + g_t(\mathbf{w})$ and $\mathbf{v} \in \partial\ell_t(\mathbf{w})$ we have that $\boldsymbol{\lambda} \in \partial g_t(\mathbf{w})$. Denote $\bar{\Delta}_t = \mathcal{D}_{t+1}(\boldsymbol{\lambda}_1,\ldots,\boldsymbol{\lambda}_{t-1},\boldsymbol{\lambda}) - \mathcal{D}_t(\boldsymbol{\lambda}_1,\ldots,\boldsymbol{\lambda}_{t-1})$. Simple algebraic manipulations yield

$$
\begin{aligned}
\bar{\Delta}_t &= -\frac{1}{2\sigma_{1:t}}\left\|\boldsymbol{\lambda}_{1:(t-1)}+\boldsymbol{\lambda}\right\|^2 + \frac{1}{2\sigma_{1:(t-1)}}\left\|\boldsymbol{\lambda}_{1:(t-1)}\right\|^2 - g_t^\star(\boldsymbol{\lambda}) \\
&= \frac{\|\boldsymbol{\lambda}_{1:(t-1)}\|^2}{2}\left(\frac{1}{\sigma_{1:(t-1)}} - \frac{1}{\sigma_{1:t}}\right) + \langle\mathbf{w},\boldsymbol{\lambda}\rangle\,\frac{\sigma_{1:(t-1)}}{\sigma_{1:t}} - \frac{\|\boldsymbol{\lambda}\|^2}{2\sigma_{1:t}} - g_t^\star(\boldsymbol{\lambda}) \\
&= \frac{\sigma_t\|\mathbf{w}\|^2}{2}\left(1 - \frac{\sigma_t}{\sigma_{1:t}}\right) + \langle\mathbf{w},\boldsymbol{\lambda}\rangle\,\frac{\sigma_{1:(t-1)}}{\sigma_{1:t}} - \frac{\|\boldsymbol{\lambda}\|^2}{2\sigma_{1:t}} - g_t^\star(\boldsymbol{\lambda}) \\
&= \underbrace{\frac{\sigma_t\,\|\mathbf{w}\|^2}{2} + \langle\mathbf{w},\boldsymbol{\lambda}\rangle - g_t^\star(\boldsymbol{\lambda})}_{A} - \underbrace{\left(\frac{\sigma_t^2\|\mathbf{w}\|^2}{2\sigma_{1:t}} + \frac{\sigma_t\langle\mathbf{w},\boldsymbol{\lambda}\rangle}{\sigma_{1:t}} + \frac{\|\boldsymbol{\lambda}\|^2}{2\sigma_{1:t}}\right)}_{B}
\end{aligned}
$$

Since $\boldsymbol{\lambda} \in \partial g_t(\mathbf{w})$, Lemma 1 thus implies that $\langle\mathbf{w},\boldsymbol{\lambda}\rangle - g_t^\star(\boldsymbol{\lambda}) = g_t(\mathbf{w})$. Therefore, $A = \ell_t(\mathbf{w})$. Next, we note that $B = \frac{\|\sigma_t\mathbf{w}+\boldsymbol{\lambda}\|^2}{2\sigma_{1:t}}$. We have thus shown that $\bar{\Delta}_t = \ell_t(\mathbf{w}) - \frac{\|\sigma_t\mathbf{w}+\boldsymbol{\lambda}\|^2}{2\sigma_{1:t}}$. Plugging the definition of $\boldsymbol{\lambda}$ into the above we conclude our proof. ■

Combining Lemma 2 with Eq. (7) and Eq. (8) we obtain the following:

FOR $t = 1, 2, \ldots, T$:

    Define $\mathbf{w}_t = -\frac{1}{\sigma_{1:(t-1)}}\boldsymbol{\lambda}_{1:(t-1)}^t$

    Receive a function $\ell_t(\mathbf{w}) = \frac{\sigma_t}{2}\|\mathbf{w}\|^2 + g_t(\mathbf{w})$ and suffer loss $\ell_t(\mathbf{w}_t)$

    Update $\boldsymbol{\lambda}_1^{t+1}, \ldots, \boldsymbol{\lambda}_t^{t+1}$ s.t. the following holds

        $\exists \boldsymbol{\lambda}_t \in \partial g_t(\mathbf{w}_t),\ \text{s.t.}\ \mathcal{D}_{t+1}(\boldsymbol{\lambda}_1^{t+1}, \ldots, \boldsymbol{\lambda}_t^{t+1}) \geq \mathcal{D}_{t+1}(\boldsymbol{\lambda}_1^t, \ldots, \boldsymbol{\lambda}_{t-1}^t, \boldsymbol{\lambda}_t)$

Figure 2: A primal-dual algorithmic framework for online convex optimization. Again, $\mathbf{w}_1 = 0$.

**Corollary 1** *Let $\ell_1, \ldots, \ell_T$ be a sequence of functions such that for all $t \in [T]$, $\ell_t$ is $\sigma_t$-strongly convex. Assume that the FTL algorithm runs on this sequence and for each $t \in [T]$, let $\mathbf{v}_t$ be in $\partial \ell_t(\mathbf{w}_t)$. Then,*

$$\sum_{t=1}^T \ell_t(\mathbf{w}_t) - \min_{\mathbf{w}} \sum_{t=1}^T \ell_t(\mathbf{w}) \ \leq \ \frac{1}{2}\sum_{t=1}^T \frac{\|\mathbf{v}_t\|^2}{\sigma_{1:t}} \tag{9}$$

*Furthermore, let $L = \max_t \|\mathbf{v}_t\|$ and assume that for all $t \in [T]$, $\sigma_t \geq \sigma$. Then, the regret is bounded by $\frac{L^2}{2\sigma}(\log(T) + 1)$.*

If we are dealing with the square loss $\ell_t(\mathbf{w}) = \|\mathbf{w} - \boldsymbol{\mu}_t\|_2^2$ (where nature chooses $\boldsymbol{\mu}_t$), then it is straightforward to see that Eq. (8) holds with equality, and this leads to the previous regret bound holding with equality. This equality is the underlying reason that the FTL strategy is a minimax strategy (See Abernethy et al. [2008] for a proof of this claim).

## 4 A Primal-Dual Algorithm for Online Strongly Convex Optimization

In the previous section, we provided a dual analysis for FTL. Here, we extend the analysis and derive a more general algorithmic framework for online optimization.

We start by examining the analysis of the FTL algorithm. We first make the important observation that Lemma 2 is not specific to the FTL algorithm and in fact holds for any configuration of dual variables. Consider an arbitrary sequence of dual variables: $(\boldsymbol{\lambda}_1^2), (\boldsymbol{\lambda}_1^3, \boldsymbol{\lambda}_2^3), \ldots, (\boldsymbol{\lambda}_1^{T+1}, \ldots, \boldsymbol{\lambda}_T^{T+1})$ and denote $\Delta_t$ as in Eq. (6). Using weak duality, we can replace the equality in Eq. (7) with the following inequality that holds for any sequence of dual variables:

$$\sum_{t=1}^T \Delta_t \ = \ \mathcal{D}_{T+1}(\boldsymbol{\lambda}_1^{T+1}, \ldots, \boldsymbol{\lambda}_T^{T+1}) \ \leq \ \min_{\mathbf{w}} \mathcal{P}_{T+1}(\mathbf{w}) \ = \ \min_{\mathbf{w}} \sum_{t=1}^T \ell_t(\mathbf{w}) \,. \tag{10}$$

A summary of the algorithmic framework is given in Fig. 2.

The following theorem, a direct corollary of the previous equation and Lemma 2, shows that all instances of the framework achieve logarithmic regret.

**Theorem 1** *Let $\ell_1, \ldots, \ell_T$ be a sequence of functions such that for all $t \in [T]$, $\ell_t$ is $\sigma_t$-strongly convex. Then, any algorithm that can be derived from Fig. 2 satisfies*

$$\sum_{t=1}^T \ell_t(\mathbf{w}_t) - \min_{\mathbf{w}} \sum_{t=1}^T \ell_t(\mathbf{w}) \ \leq \ \frac{1}{2}\sum_{t=1}^T \frac{\|\mathbf{v}_t\|^2}{\sigma_{1:t}} \tag{11}$$

*where $\mathbf{v}_t \in \partial \ell_t(\mathbf{w}_t)$.*

**Proof** Let $\Delta_t$ be as defined in Eq. (6). The last condition in the algorithm implies that

$$\Delta_t \ \geq \ \mathcal{D}_{t+1}(\boldsymbol{\lambda}_1^t, \ldots, \boldsymbol{\lambda}_{t-1}^t, \mathbf{v}_t - \sigma_t \mathbf{w}_t) - \mathcal{D}_t(\boldsymbol{\lambda}_1^t, \ldots, \boldsymbol{\lambda}_{t-1}^t) \,.$$

The proof follows directly by combining the above with Eq. (10) and Lemma 2. ∎

We conclude this section by deriving several algorithms from the framework.

**Example 1 (Follow-The-Leader)** *As we have shown in Sec. 3, the FTL algorithm (Fig. 1) is equivalent to optimizing the dual variables at each online round. This update clearly satisfies the condition in Fig. 2 and is therefore a special case.*

**Example 2 (Gradient-Descent)** *Following Hazan et al. [2006], Bartlett et al. [2007] suggested the following update rule for differentiable strongly convex function*

$$\mathbf{w}_{t+1} = \mathbf{w}_t - \frac{1}{\sigma_{1:t}} \nabla \ell_t(\mathbf{w}_t) \ . \tag{12}$$

*Using a simple inductive argument, it is possible to show that the above update rule is equivalent to the following update rule of the dual variables*

$$(\boldsymbol{\lambda}_1^{t+1}, \ldots, \boldsymbol{\lambda}_t^{t+1}) \ = \ (\boldsymbol{\lambda}_1^t, \ldots, \boldsymbol{\lambda}_{t-1}^t, \nabla \ell_t(\mathbf{w}_t) - \sigma_t \mathbf{w}_t) \ . \tag{13}$$

*Clearly, this update rule satisfies the condition in Fig. 2. Therefore our framework encompasses this algorithm as a special case.*

**Example 3 (Online Coordinate-Dual-Ascent)** *The FTL and the Gradient-Descent updates are two extreme cases of our algorithmic framework. The former makes the largest possible increase of the dual while the latter makes the smallest possible increase that still satisfies the sufficient dual increase requirement. Intuitively, the FTL method should have smaller regret as it consumes more of its potential earlier in the optimization process. However, its computational complexity is large as it requires a full blown optimization procedure at each online round. A possible compromise is to fully optimize the dual objective but only with respect to a small number of dual variables. In the extreme case, we optimize only with respect to the last dual variable. Formally, we let*

$$\boldsymbol{\lambda}_i^{t+1} \ = \ \begin{cases} \boldsymbol{\lambda}_i^t & \text{if } i < t \\ \underset{\boldsymbol{\lambda}_t}{\operatorname{argmax}} \ \mathcal{D}_{t+1}(\boldsymbol{\lambda}_1^t, \ldots, \boldsymbol{\lambda}_{t-1}^t, \boldsymbol{\lambda}_t) & \text{if } i = t \end{cases}$$

*Clearly, the above update satisfies the requirement in Fig. 2 and therefore enjoys a logarithmic regret bound. The computational complexity of performing this update is often small as we optimize over a single dual vector. Occasionally, it is possible to obtain a closed-form solution of the optimization problem and in these cases the computational complexity of the coordinate-dual-ascent update is identical to that of the gradient-descent method.*

## 5 Generalized strongly convex functions

In this section, we extend our algorithmic framework to deal with generalized strongly convex functions. We first need the following generalized definition of strong convexity.

**Definition 1** *A continuous function $f$ is $\sigma$-strongly convex over a convex set $S$ with respect to a norm $\|\cdot\|$ if $S$ is contained in the domain of $f$ and for all $\mathbf{v}, \mathbf{u} \in S$ and $\alpha \in [0,1]$ we have*

$$f(\alpha \, \mathbf{v} + (1-\alpha) \, \mathbf{u}) \ \leq \ \alpha \, f(\mathbf{v}) + (1-\alpha) \, f(\mathbf{u}) - \frac{\sigma}{2} \, \alpha \, (1-\alpha) \, \|\mathbf{v} - \mathbf{u}\|^2 \ . \tag{14}$$

It is straightforward to show that the function $f(\mathbf{w}) = \frac{1}{2}\|\mathbf{w}\|_2^2$ is strongly convex with respect to the Euclidean norm. Less trivial examples are given below.

**Example 4** *The function $f(\mathbf{w}) = \sum_{i=1}^n w_i \log(w_i/\frac{1}{n})$ is strongly convex over the probability simplex, $S = \{\mathbf{w} \in \mathbb{R}_+^n : \|\mathbf{w}\|_1 = 1\}$, with respect to the $L_1$ norm. Its conjugate function is $f^\star(\boldsymbol{\theta}) = \log(\frac{1}{n} \sum_{i=1}^n \exp(\theta_i))$.*

**Example 5** *For $q \in (1, 2)$, the function $f(\mathbf{w}) = \frac{1}{2(q-1)}\|\mathbf{w}\|_q^2$ is strongly convex over $S = \mathbb{R}^n$ with respect to the $L_q$ norm. Its conjugate function is $f^\star(\boldsymbol{\theta}) = \frac{1}{2(p-1)}\|\boldsymbol{\theta}\|_p^2$, where $p = (1 - 1/q)^{-1}$.*

For proofs, see for example Shalev-Shwartz [2007]. In the appendix, we list several important properties of strongly convex functions. In particular, the Fenchel conjugate of a strongly convex function is differentiable.

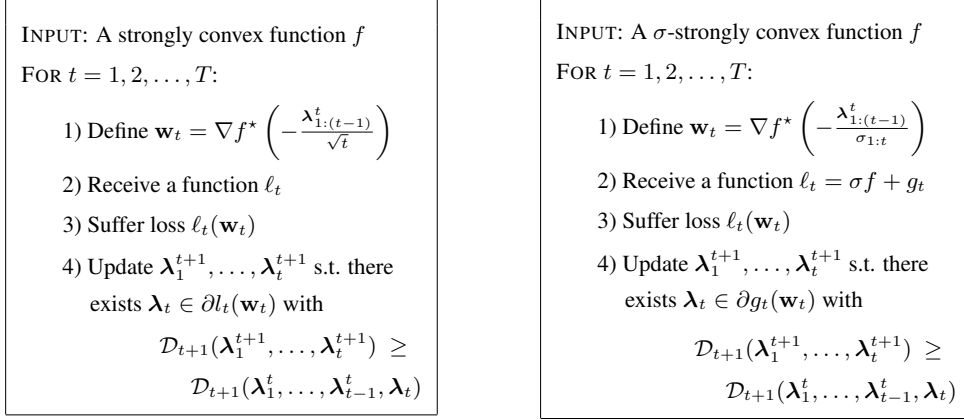

Figure 3: Primal-dual template algorithms for general online convex optimization (left) and online strongly convex optimization (right). Here $a_{1:t} = \sum_{i=1}^{t} a_i$, and for notational convenient, we implicitly assume that $a_{1:0} = 0$. See text for description.

Consider the case where for all $t$, $\ell_t$ can be written as $\sigma_t f + g_t$ where $f$ is 1-strongly convex with respect to some norm $\|\cdot\|$ and $g_t$ is a convex function. We also make the simplifying assumption that $\sigma_t$ is known to the forecaster before he defines $\mathbf{w}_t$.

For each round $t$, we now define the primal objective to be

$$\mathcal{P}_t(\mathbf{w}) \;=\; \sigma_{1:(t-1)} f(\mathbf{w}) + \sum_{i=1}^{t-1} g_i(\mathbf{w}) \;. \tag{15}$$

The dual objective is (see again Sec. 2)

$$\mathcal{D}_t(\boldsymbol{\lambda}_1, \ldots, \boldsymbol{\lambda}_{t-1}) \;=\; -\sigma_{1:(t-1)} f^\star \left( -\frac{\boldsymbol{\lambda}_{1:(t-1)}}{\sigma_{1:(t-1)}} \right) - \sum_{i=1}^{t-1} g_i^\star(\boldsymbol{\lambda}_i) \;. \tag{16}$$

An algorithmic framework for online optimization in the presence of general strongly convex functions is given on the right-hand side of Fig. 3.

The following theorem provides a logarithmic regret bound for the algorithmic framework given on the right-hand side of Fig. 3.

**Theorem 2** *Let $\ell_1, \ldots, \ell_T$ be a sequence of functions such that for all $t \in [T]$, $\ell_t = \sigma_t f + g_t$ for $f$ being strongly convex w.r.t. a norm $\|\cdot\|$ and $g_t$ is a convex function. Then, any algorithm that can be derived from Fig. 3 (right) satisfies*

$$\sum_{t=1}^{T} \ell_t(\mathbf{w}_t) - \min_{\mathbf{w}} \sum_{t=1}^{T} \ell_t(\mathbf{w}) \;\leq\; \frac{1}{2} \sum_{t=1}^{T} \frac{\|\mathbf{v}_t\|_\star^2}{\sigma_{1:t}} \;, \tag{17}$$

*where $\mathbf{v}_t \in \partial g_t(\mathbf{w}_t)$ and $\|\cdot\|_\star$ is the norm dual to $\|\cdot\|$.*

The proof of the theorem is given in Sec. B

## 6  Summary

In this paper, we extended the primal-dual algorithmic framework for general convex functions from Shalev-Shwartz and Singer [2006], Shalev-Shwartz [2007] to strongly convex functions. The template algorithms are outlined in Fig. 3. The left algorithm is the primal-dual algorithm for general convex functions from Shalev-Shwartz and Singer [2006], Shalev-Shwartz [2007]. Here, $f$ is the complexity function, $(\boldsymbol{\lambda}_1^t, \ldots, \boldsymbol{\lambda}_t^t)$ are the dual variables at time $t$, and $D_t(\cdot)$ is the dual objective

function at time $t$ (which is a lower bound on primal value). The function $\nabla f^\star$ is the gradient of the conjugate function of $f$, which can be viewed as a projection of the dual variables back into the primal space. At the least aggressive extreme, in order to obtain $\sqrt{T}$ regret, it is sufficient to set $\boldsymbol{\lambda}_t^i$ (for all $i$) to be a subgradient of the loss $\partial \ell_t(\mathbf{w}_t)$. We then recover an online 'mirror descent' algorithm [Beck and Teboulle, 2003, Grove et al., 2001, Kivinen and Warmuth, 1997], which specializes to gradient descent when $f$ is the squared 2-norm or the exponentiated gradient descent algorithm when $f$ is the relative entropy. At the most aggressive extreme, where $\mathcal{D}_t$ is maximized at each round, we have 'Follow the Regularized Leader', which is $\mathbf{w}_t = \arg\min_{\mathbf{w}} \sum_{i=1}^{t-1} \ell_i(\mathbf{w}) + \sqrt{t} f(w)$. Intermediate algorithms can also be devised such as the 'passive-aggressive' algorithms [Crammer et al., 2006, Shalev-Shwartz, 2007].

The right algorithm in Figure 3 is our new contribution for strongly convex functions. Any $\sigma$-strongly convex loss function can be decomposed into $\ell_t = \sigma f + g_t$, where $g_t$ is convex. The algorithm for strongly convex functions is different in two ways. First, the effective learning rate is now $\frac{1}{\sigma_{1:t}}$ rather than $\frac{1}{\sqrt{t}}$ (see Step 1 in both algorithms). Second, more subtly, the condition on the dual variables (in Step 4) is only determined by the subgradient of $g_t$, rather than the subgradient of $\ell_t$. At the most aggressive end of the spectrum, where $\mathcal{D}_t$ is maximized at each round, we have the 'Follow the Leader' (FTL) algorithm: $\mathbf{w}_t = \arg\min_{\mathbf{w}} \sum_{i=1}^{t-1} \ell_i(\mathbf{w})$. At the least aggressive end, we have the gradient descent algorithm of Hazan et al. [2006] (which uses learning rate $\frac{1}{\sigma_{1:t}}$). Furthermore, we provide algorithms which lie in between these two extremes — it is these algorithms which have the potential for most practical impact.

Empirical observations suggest that algorithms which most aggressively close the duality gap tend to perform most favorably [Crammer et al., 2006, Shalev-Shwartz and Singer, 2007b]. However, at the FTL extreme, this is often computationally prohibitive to implement (as one must solve a full blown optimization problem at each round). Our template algorithm suggests a natural compromise, which is to optimize the dual objective but only with respect to a small number of dual variables (say the most current dual variable) — we coin this algorithm online coordinate-dual-ascent. In fact, it is sometimes possible to obtain a closed-form solution of this optimization problem, so that the computational complexity of the coordinate-dual-ascent update is identical to that of a vanilla gradient-descent method. This variant update still enjoys a logarithmic regret bound.

# References

J. Abernethy, P. Bartlett, A. Rakhlin, and A. Tewari. Optimal strategies and minimax lower bounds for online convex games. In *Proceedings of the Nineteenth Annual Conference on Computational Learning Theory*, 2008.

P. L. Bartlett, E. Hazan, and A. Rakhlin. Adaptive online gradient descent. In *Advances in Neural Information Processing Systems 21*, 2007.

A. Beck and M. Teboulle. Mirror descent and nonlinear projected subgradient methods for convex optimization. *Operations Research Letters*, 31:167–175, 2003.

J. Borwein and A. Lewis. *Convex Analysis and Nonlinear Optimization*. Springer, 2006.

S. Boyd and L. Vandenberghe. *Convex Optimization*. Cambridge University Press, 2004.

N. Cesa-Bianchi and G. Lugosi. *Prediction, learning, and games*. Cambridge University Press, 2006.

M. Collins. Discriminative training methods for hidden Markov models: Theory and experiments with perceptron algorithms. In *Conference on Empirical Methods in Natural Language Processing*, 2002.

K. Crammer, O. Dekel, J. Keshet, S. Shalev-Shwartz, and Y. Singer. Online passive aggressive algorithms. *Journal of Machine Learning Research*, 7:551–585, Mar 2006.

A. J. Grove, N. Littlestone, and D. Schuurmans. General convergence results for linear discriminant updates. *Machine Learning*, 43(3):173–210, 2001.

E. Hazan, A. Kalai, S. Kale, and A. Agarwal. Logarithmic regret algorithms for online convex optimization. In *Proceedings of the Nineteenth Annual Conference on Computational Learning Theory*, 2006.

J. Kivinen and M. Warmuth. Exponentiated gradient versus gradient descent for linear predictors. *Information and Computation*, 132(1):1–64, January 1997.

S. Shalev-Shwartz. *Online Learning: Theory, Algorithms, and Applications*. PhD thesis, The Hebrew University, 2007.

S. Shalev-Shwartz and Y. Singer. Convex repeated games and Fenchel duality. In *Advances in Neural Information Processing Systems 20*, 2006.

S. Shalev-Shwartz and Y. Singer. Logarithmic regret algorithms for strictly convex repeated games. Technical report, The Hebrew University, 2007a. Available at http://www.cs.huji.ac.il/∼shais.

S. Shalev-Shwartz and Y. Singer. A unified algorithmic approach for efficient online label ranking. In *aistat07*, 2007b.

